# A Constructive RBF Network for Writer Adaptation

**John C. Platt and Nada P. Matić**
Synaptics, Inc.
2698 Orchard Parkway
San Jose, CA 95134
platt@synaptics.com, nada@synaptics.com

## Abstract

This paper discusses a fairly general adaptation algorithm which augments a standard neural network to increase its recognition accuracy for a specific user. The basis for the algorithm is that the output of a neural network is characteristic of the input, even when the output is incorrect. We exploit this characteristic output by using an *Output Adaptation Module* (OAM) which maps this output into the correct user-dependent confidence vector. The OAM is a simplified Resource Allocating Network which constructs radial basis functions on-line. We applied the OAM to construct a writer-adaptive character recognition system for on-line hand-printed characters. The OAM decreases the word error rate on a test set by an average of 45%, while creating only 3 to 25 basis functions for each writer in the test set.

## 1 Introduction

One of the major difficulties in creating any statistical pattern recognition system is that the statistics of the training set is often different from the statistics in actual use. The creation of a statistical pattern recognizer is often considered as a regression problem, where class probabilities are estimated from a fixed training set. Statistical pattern recognizers tend to work well for typical data that is similar to the training set data, but do not work well for atypical data that is not well represented in the training set. Poor performance on atypical data is a problem for human interfaces, because people tend to provide drastically non-typical data (for example, see figure 1).

The solution to this difficulty is to create an adaptive recognizer, instead of treating recognition as a static regression problem. The recognizer must adapt to new statistics during use. As applied to on-line handwriting recognition, an adaptive

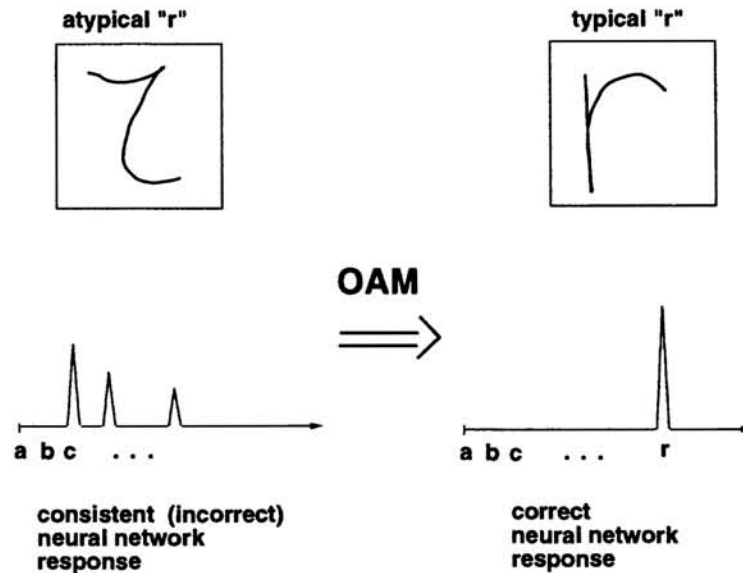

Figure 1: When given atypical input data, the neural network produces a consistent incorrect output pattern. The OAM recognizes the consistent pattern and produces a corrected user-adaptive output.

recognizer improves the accuracy for a particular user by adapting the recognizer to that user.

This paper proposes a novel method for creating an adaptive recognizer, which we call the *Output Adaptation Module* or OAM. The OAM was inspired by the development of a writer-independent neural network handwriting recognizer. We noticed that the output of this neural network was characteristic of the input: if a specific style of character was shown to the network, the network's output was almost always consistent for that specific style, even when the output was incorrect.

To exploit the consistency of the incorrect outputs, we decided to add an OAM on top of the network. The OAM learns to recognize these consistent incorrect output vectors, and produces a more correct output vector (see figure 1). The units of the OAM are radial basis functions (RBF) [5]. Adaptation of these RBF units is performed using a simplified version of the Resource Allocating Network (RAN) algorithm of Platt [4][2]. The number of units that RAN allocates scales sub-linearly with the number of presented learning examples, in contrast to other algorithms which allocate a new unit for every learned example.

The OAM has the following properties, which are useful for a user-adaptive recognizer:

- The adaptation is very fast: the user need only provide a few additional examples of his own data.
- There is very little recognition speed degradation.
- A modest amount of additional memory per user is required.
- The OAM is not limited to neural network recognizers.
- The output of the OAM is a corrected vector of confidences, which is more useful for contextual post-processing than a single label.

## 1.1 Relationship to Previous Work

The OAM is related to previous work in user adaptation of neural recognizers for both speech and handwriting.

A previous example of user adaptation of a neural handwriting recognizer employed a Time Delay Neural Network (TDNN), where the last layer of a TDNN was replaced with a tunable classifier that is more appropriate for adaptation [1][3]. In Guyon, et al. [1], the last layer of a TDNN was replaced by a $k$-nearest neighbor classifier. This work was further extended in Matić, et al. [3], where the last layer of the TDNN was replaced with an optimal hyperplane classifier which is retrained for adaptation purposes. The optimal hyperplane classifier retained the same accuracy as the $k$-nearest neighbor classifier, while reducing the amount of computation and memory required for adaptation.

The present work improves upon these previous user-adaptive handwriting systems in three ways. First, the OAM does not require the retraining and storage of the entire last layer of the network. The OAM thus further reduces both CPU and memory requirements. Second, the OAM produces an output vector of confidences, instead of simply an output label. This vector of confidences can be used effectively by a contextual post-processing step, while a label cannot. Third, our adaptation experiments are performed on a neural network which recognizes a full character set. These previous papers only experimented with neural networks that recognized character subsets, which is a less difficult adaptation problem.

The OAM is related to stacking [6]. In stacking, outputs of multiple recognizers are combined via training on partitions of the training set. With the OAM, the multiple outputs of a recognizer are combined using memory-based learning. The OAM is trained on the idiosyncratic statistics of actual use, not on a pre-defined training set partition.

## 2 The Output Adaptation Module (OAM)

Section 2 of this paper describes the OAM in detail, while section 3 describes its application to create a user-adaptive handwriting recognizer.

The OAM maps the output of a neural network $V_i$ into a user-adapted output $O_i$, by adding an adaptation vector $A_i$:

$$O_i = V_i + A_i. \tag{1}$$

Depending on the neural network training algorithm used, both the output of the neural network $V_i$ and the user-adapted output $O_i$ can estimate *a posteriori* class probabilities, suitable for further post-processing.

The goal of the OAM is to bring the output $O_i$ closer to an ideal response $T_i$. In our experiments, the target $T_i$ is 0.9 for the neuron corresponding to the correct character and 0.1 for all other neurons.

The adaptation vector $A_i$ is computed by a radial basis function network that takes $V_i$ as an input:

$$O_i = V_i + A_i \;\; = \;\; V_i + \sum_j C_{ij} \Phi_j(\vec{V}), \tag{2}$$

$$\Phi_j(\vec{V}) \;\; = \;\; f\left(\frac{d(\vec{V}, \vec{M}_j)}{R_j}\right). \tag{3}$$

where $\vec{M}_j$ is the center of the $j$th radial basis function, $d$ is a distance metric between $\vec{V}$ and $\vec{M}_j$, $R_j$ is a parameter that controls the width of the $j$th basis function, $f$ is

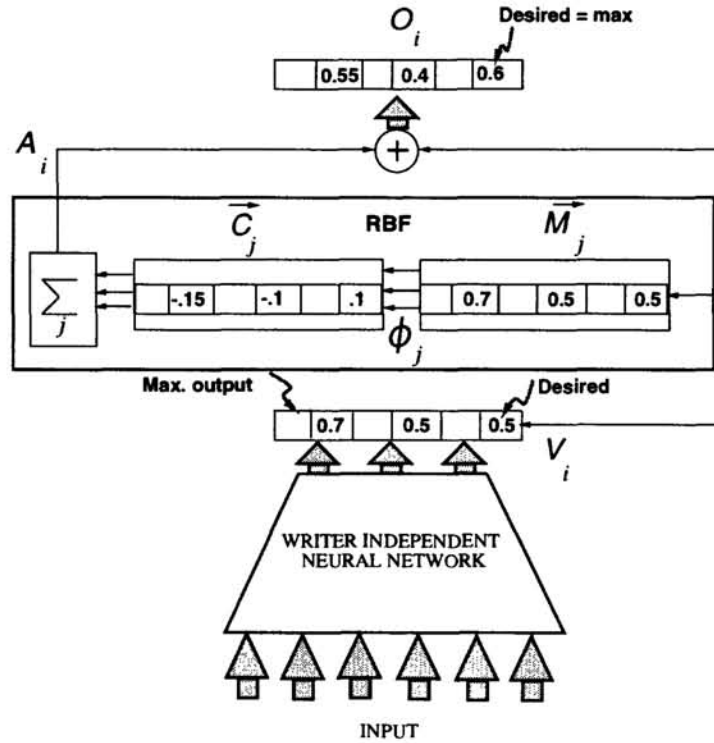

Figure 2: The architecture of the OAM.

a decreasing function that controls the shape of the basis functions, and $C_{ij}$ is the amount of correction that the $j$th basis function adds to the output. We call $\vec{M}_j$ the memories in the adaptation module, and we call $\vec{C}_j$ the correction vectors (see figure 2).

The function $f$ is a decreasing polynomial function:

$$f(x) = \begin{cases} \left(1 - x^2\right)^2, & \text{if } x < 1; \\ 0, & \text{otherwise.} \end{cases} \quad (4)$$

The distance function $d$ is a Euclidean distance metric that first clips both of its input vectors to the range $[0.1, 0.9]$ in order to reduce spurious noise.

The algorithm for constructing the radial basis functions is a simplification of the RAN algorithm [4][2]. The OAM starts with no memories or corrections. When the user corrects a recognition error, the OAM finds the distance $d_{\min}$ from the nearest memory to the vector $V_i$. If the distance $d_{\min}$ is greater than a threshold $\delta$, then a new RBF unit is allocated, with a new memory that is set to the vector $V_i$, and a corresponding correction vector that is set to correct the error with a step size $a$:

$$C_{ik} = a(T_i - O_i). \quad (5)$$

If the distance $d_{\min}$ is less than $\delta$, no new unit is allocated: the correction vector of the nearest memory to $V_i$ is updated to correct the error by a step size $b$.

$$\Delta C_{ik} = b(T_i - O_i)\Phi_k(\vec{V}). \quad (6)$$

For our experiments, we set $\delta = 0.1$, $a = 0.25$, and $b = 0.2$. The values $a$ and $b$ are chosen to be less than 1 to sacrifice learning speed to gain learning stability.

The number of radial basis functions grows sub-linearly with the number of errors, because units are only allocated for novel errors that the OAM has not seen before.

For errors similar to those the OAM has seen before, the algorithm updates one of the correction vectors using a simplified LMS rule (eq. 6).

In the computation of the nearest memory, we always consider an additional phantom memory: the target $\vec{Q}$ that corresponds to the highest output in $\vec{V}$. This phantom memory is considered in order to prevent the OAM from allocating memories when the neural network output is unambiguous. The phantom memory prevents the OAM from affecting the output for neatly written characters.

The adaptation algorithm used is described as pseudo-code, below:

```
For every character shown to the network {
    If the user indicates an error {
        T = target vector of the correct character
        Q = target vector of the highest element in V
        d_min = min(min_j d(V, M_j), d(V, Q))
        If d_min > δ {   // allocate a new memory
            k = index of the new memory
            C_ik = a(T_i − O_i)
            M_k = V
            R_j = d_min
        }
        else if memories exist and min_j d(V, M_j) < d(V, Q) {
            k = arg min_j d(V, M_j)
            C_ik = C_ik + b(T_i − O_i)Φ_k(V)
        }
    }
}
```

## 3  Experiments and Results

To test the effectiveness of the OAM, we used it to create a writer-adaptive handwriting recognition system. The OAM was connected to the outputs of a writer-independent neural network trained to recognize characters hand-printed in boxes. This neural network was a carefully tuned multilayer feed-forward network, trained with the back-propagation algorithm. The network has 510 inputs, 200 hidden units, and 72 outputs.

The input to the OAM was a vector of 72 confidences, one per class. These confidences were in the range $[0, 1]$. There is one input for every upper case character, lower case character, and digit. There is also one input for each member of a subset of punctuation characters ( !$&',-:;=? ).

The OAM was tested interactively. Tests of the OAM were performed by five writers disjoint from the training writers for the writer-independent neural network. These writers had atypical writing styles which were difficult for the network to recognize. Test characters were entered word-by-word on a tablet. The writers were instructed to write more examples of characters that reflected their atypical writing style. The words that these writers used were not taken from a word list, and could consist of any combination of the 72 available characters. Users were shown the results of the OAM, combined into words and further processed by a dictionary. Whenever the system failed to recognize a word correctly, all misclassified characters and their corresponding desired labels were used by the OAM to adapt the system to the

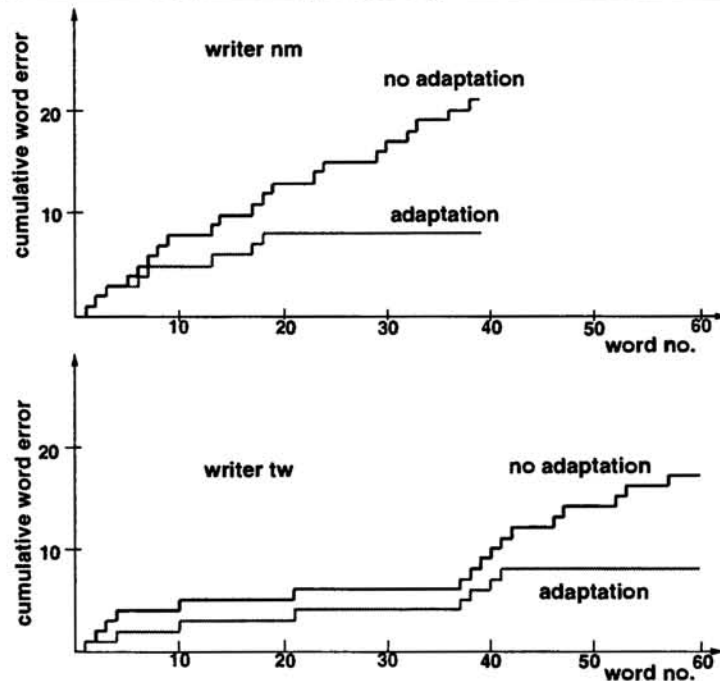

Figure 3: The cumulative number of word errors for the writer "nm" and the writer "tw", with and without adaptation

| Writer | % word error no OAM | % word error OAM | Memories stored during test | Words written during test |
|--------|---------------------|------------------|-----------------------------|---------------------------|
| rag    | 25%                 | 17%              | 6                           | 93                        |
| mfe    | 62%                 | 36%              | 12                          | 53                        |
| qm     | 42%                 | 31%              | 25                          | 80                        |
| nm     | 54%                 | 21%              | 4                           | 39                        |
| tw     | 28%                 | 10%              | 3                           | 60                        |

Table 1: Quantitative test results for the OAM.

particular user.

Figure 3 shows the performance of the OAM for the writer "nm" and for the writer "tw". The total number of word errors since adaptation started is plotted against the number of words shown to the OAM. The baseline cumulative error without the OAM is also given. The slope of each curve gives the estimate of the instantaneous error rate. The OAM causes the slope for both writers to decrease dramatically as the adaptation progresses. Over the last third of the test set, the word error rate for writer "nm" was 0%, while the word error rate for writer "tw" was 5%. These examples show the the OAM can substantially improve the accuracy of a writer-independent neural network.

Quantitative results are shown in table 1, where the word error rates obtained with the OAM are compared to the baseline word error rates without the OAM. The right two columns contain the number of stored basis functions and the number of words tested by the OAM.

The OAM corrects an average of 45% of the errors in the test set. The accuracy

rates with the OAM were taken for the entire test run, and therefore count the errors that were made while adaptation was taking place. By the end of the test, the true error rates for these writers would be even lower than those shown in table 1, as can be seen in figure 3.

These experiments showed that the OAM adapts very quickly and requires a small amount of additional memory and computation. For most writers, only 2-3 presentations of a writer-dependent variant of a character were sufficient for the OAM to adapt. The maximum number of stored basis functions were 25 in these experiments. The OAM did not substantially affect the recognition speed of the system.

## 4 Conclusions

We have designed a widely applicable *Output Adaptation Module* (OAM) to place on top of standard neural networks. The OAM takes the output of the network as input, and determines an additional adaptation vector to add to the output. The adaptation vector is computed via a radial basis function network, which is learned with a simplification of the RAN algorithm. The OAM has many nice properties: only a few examples are needed to learn atypical inputs, the number of stored memories grows sub-linearly with the number of errors, the recognition rate of the writer-independent neural network is unaffected by adaptation and the output of the module is a confidence vector suitable for further post-processing.

The OAM addresses the difficult problem of creating a high-perplexity adaptive recognizer. We applied the OAM to create a writer-adaptive handwriting recognition system. On a test set of five difficult writers, the adaptation module decreased the error rate by 45%, and only stored between 3 and 25 basis functions per writer.

## 5 Acknowledgements

We wish to thank Steve Nowlan for helpful suggestions during the development of the OAM algorithm and for his work on the writer-independent neural network. We would also like to thank Joe Decker for his work on the writer-independent neural network.

## References

[1] I. Guyon, D. Henderson, P. Albrecht, Y. Le Cun, and J. Denker. Writer independent and writer adaptive neural network for on-line character recognition. In S. Impedovo, editor, *From Pixels to Features III*, Amsterdam, 1992. Elsevier.

[2] V. Kadirkamanathan and M. Niranjan. A function estimation approach to sequential learning with neural networks. *Neural Computation*, 5(6):954–976, 1993.

[3] N. Matić, I. Guyon, J. Denker, and V. Vapnik. Writer adaptation for on-line handwritten character recognition. In *ICDAR93*, Tokyo, 1993. IEEE Computer Society Press.

[4] J. Platt. A resource-allocating network for function interpolation. *Neural Computation*, 3(2):213–225, 1991.

[5] M. Powell. Radial basis functions for multivariate interpolation: A review. In J. C. Mason and M. G. Cox, editors, *Algorithms for Approximation*, Oxford, 1987. Clarendon Press.

[6] D. Wolpert. Stacked generalization. *Neural Networks*, 5(2):241–260, 1992.